# Recognizing Hand-written Digits Using Hierarchical Products of Experts

**Guy Mayraz & Geoffrey E. Hinton**
Gatsby Computational Neuroscience Unit
University College London
17 Queen Square, London WC1N 3AR, U.K.

## Abstract

The product of experts learning procedure [1] can discover a set of stochastic binary features that constitute a non-linear generative model of handwritten images of digits. The quality of generative models learned in this way can be assessed by learning a separate model for each class of digit and then comparing the unnormalized probabilities of test images under the 10 different class-specific models. To improve discriminative performance, it is helpful to learn a hierarchy of separate models for each digit class. Each model in the hierarchy has one layer of hidden units and the $n^{th}$ level model is trained on data that consists of the activities of the hidden units in the already trained $(n-1)^{th}$ level model. After training, each level produces a separate, unnormalized log probabilty score. With a three-level hierarchy for each of the 10 digit classes, a test image produces 30 scores which can be used as inputs to a supervised, logistic classification network that is trained on separate data. On the MNIST database, our system is comparable with current state-of-the-art discriminative methods, demonstrating that the product of experts learning procedure can produce effective generative models of high-dimensional data.

## 1 Learning products of stochastic binary experts

Hinton [1] describes a learning algorithm for probabilistic generative models that are composed of a number of experts. Each expert specifies a probability distribution over the visible variables and the experts are combined by multiplying these distributions together and renormalizing.

$$p(\mathbf{d}|\theta_1...\theta_n) = \frac{\Pi_m p_m(\mathbf{d}|\theta_m)}{\sum_{\mathbf{c}} \Pi_m p_m(\mathbf{c}|\theta_m)} \quad (1)$$

where $\mathbf{d}$ is a data vector in a discrete space, $\theta_m$ is all the parameters of individual model $m$, $p_m(\mathbf{d}|\theta_m)$ is the probability of $\mathbf{d}$ under model $m$, and $\mathbf{c}$ is an index over all possible vectors in the data space.

A Restricted Boltzmann machine [2, 3] is a special case of a product of experts in which each expert is a single, binary stochastic hidden unit that has symmetrical connections to a set of visible units, and connections between the hidden units are forbidden. Inference in an RBM is much easier than in a general Boltzmann machine and it is also much easier

than in a causal belief net because there is no explaining away. There is therefore no need to perform any iteration to determine the activities of the hidden units. The hidden states, $s_j$, are conditionally independent given the visible states, $s_i$, and the distribution of $s_j$ is given by the standard logistic function:

$$p(s_j = 1) = \frac{1}{1 + \exp(-\sum_i w_{ij} s_i)} \qquad (2)$$

Conversely, the hidden states of an RBM are *marginally* dependent so it is easy for an RBM to learn population codes in which units may be highly correlated. It is hard to do this in causal belief nets with one hidden layer because the generative model of a causal belief net assumes marginal independence.

An RBM can be trained using the standard Boltzmann machine learning algorithm which follows a noisy but unbiased estimate of the gradient of the log likelihood of the data. One way to implement this algorithm is to start the network with a data vector on the visible units and then to alternate between updating all of the hidden units in parallel and updating all of the visible units in parallel. Each update picks a binary state for a unit from its posterior distribution given the current states of all the units in the other set. If this alternating Gibbs sampling is run to equilibrium, there is a very simple way to update the weights so as to minimize the Kullback-Leibler divergence, $Q^0 \| Q^\infty$, between the data distribution, $Q^0$, and the equilibrium distribution of fantasies over the visible units, $Q^\infty$, produced by the RBM [4]:

$$\Delta w_{ij} \propto <s_i s_j>_{Q^0} - <s_i s_j>_{Q^\infty} \qquad (3)$$

where $<s_i s_j>_{Q^0}$ is the expected value of $s_i s_j$ when data is clamped on the visible units and the hidden states are sampled from their conditional distribution given the data, and $<s_i s_j>_{Q^\infty}$ is the expected value of $s_i s_j$ after prolonged Gibbs sampling.

This learning rule does not work well because it can take a long time to approach thermal equilibrium and the sampling noise in the estimate of $<s_i s_j>_{Q^\infty}$ can swamp the gradient. [1] shows that it is far more effective to minimize the *difference* between $Q^0 \| Q^\infty$ and $Q^1 \| Q^\infty$ where $Q^1$ is the distribution of the one-step reconstructions of the data that are produced by first picking binary hidden states from their conditional distribution given the data and then picking binary visible states from their conditional distribution given the hidden states. The exact gradient of this "contrastive divergence" is complicated because the distribution $Q^1$ depends on the weights, but [1] shows that this dependence can safely be ignored to yield a simple and effective learning rule for following the approximate gradient of the contrastive divergence:

$$\Delta w_{ij} \propto <s_i s_j>_{Q^0} - <s_i s_j>_{Q^1} \qquad (4)$$

For images of digits, it is possible to apply Eq. 4 directly if we use stochastic binary pixel intensities, but it is more effective to normalize the intensities to lie in the range $[0, 1]$ and then to use these real values as the inputs to the hidden units. During reconstruction, the stochastic binary pixel intensities required by Eq. 4 are also replaced by real-valued probabilities. Finally, the learning rule can be made less noisy by replacing the stochastic binary activities of the hidden units by their expected values. So the learning rule we actually use is:

$$\Delta w_{ij} \propto <p_i p_j>_{Q^0} - <p_i p_j>_{Q^1} \qquad (5)$$

Stochastically chosen binary states of the hidden units are still used for computing the probabilities of the reconstructed pixels. This prevents each real-valued hidden probability from conveying more than 1 bit of information to the reconstruction.

## 2  The MNIST database

MNIST, a standard database for testing digit recognition algorithms, is available at http://www.research.att.com/~yann/ocr/mnist/index.html. MNIST

| METHOD | % ERRORS |
|---|---|
| Linear classifier (1-layer NN) | 12.0 |
| K-nearest-neighbors, Euclidean | 5.0 |
| 1000 RBF + linear classifier | 3.6 |
| Best Back-Prop: 3-layer NN, 500+150 hidden units | 2.95 |
| | |
| Reduced Set SVM deg 5 polynomial | 1.0 |
| LeNet-1 [with 16x16 input] | 1.7 |
| LeNet-5 | 0.95 |
| | |
| **Product of Experts** (separate 3-layer net for each model) | **1.7** |

Table 1: Performance of various learning methods on the MNIST test set.

has 60,000 training images and 10,000 test images. Images are highly variable in style but are size-normalized and translated so that the center of gravity of their intensity lies at the center of a fixed-size image of 28 by 28 pixels.

A number of well-known learning algorithms have been run on the MNIST database[5], so it is easy to assess the relative performance of a novel algorithm. Some of the experiments in [5] included deskewing images or augmenting the training set with distorted versions of the original images. We did not use deskewing or distortions in our main experiments, so we only compare our results with other methods that did not use them. The results in Table 1 should be treated with caution. Some attempts to replicate the degree 5 polynomial SVM have produced slightly higher error rates of 1.4% [6] and standard backpropagation can be carefully tuned to achieve under 2% (John Platt, personal communication).

Table 1 shows that it is possible to achieve a result that is comparable with the best discriminative techniques by using multiple PoE models of each digit class to extract scores that represent unnormalized log probabilities. These scores are then used as the inputs to a simple logistic classifier. The fact that a system based on generative models can come close to the very best discriminative systems suggests that the generative models are doing a good job of capturing the distributions.

## 3 Training the individual PoE models

The MNIST database contains an average of 6,000 training examples per digit, but these examples are unevenly distributed among the digit classes. In order to simplify the research we produced a balanced database by using only 5,400 examples of each digit. The first 4,400 examples were the *unsupervised training set* used for training the individual PoE models. The remaining examples of each of the 10 digits constituted the *supervised training set* used for training the logistic classification net that converts the scores of all the PoE models into a classification.

The original intensity range in the MNIST images was 0 to 255. This was normalized to the range 0 to 1 so that we could treat intensities as probabilities. The normalized pixel intensities were used as the initial activities of the 784 visible units corresponding to the 28 by 28 pixels. The visible units were fully connected to a single layer of hidden units. The weights between the input and hidden layer were initialized to small, zero-mean, Gaussian-distributed, random values. The 4,400 training examples were divided into 44 mini-batches. One epoch of learning consisted of a pass through all 44 minibatches in fixed order with the weights being updated after each minibatch. We used a momentum method with a small

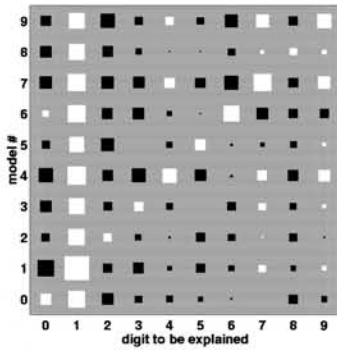

Figure 1: The areas of the blobs show the mean goodness of validation set digits using only the first-level models with 500 hidden units (white is positive). A different constant is added to all the goodness scores of each model so that rows sum to zero. Successful discrimination depends on models being better on their own class than other models are. The converse is not true: models can be better reconstructing other, easier classes of digits than their own class.

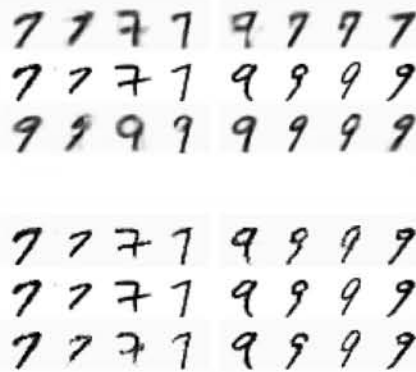

Figure 2: Cross reconstruction of 7s and 9s with models containing 25 hidden units (top) and 100 hidden units (bottom). The central horizontal line in each block contains originals, and the lines above and below are reconstructions by the 7s and 9s models respectively. Both models produce stereotyped digits in the small net and much better reconstructions in the large one for both the digit classes. The 9s model sometimes tries to close the loop in 7s, and the 7s model tries to open the loop in 9s.

amount of weight decay, so the change in a weight after the $t^{th}$ minibatch was:

$$\Delta w_{ij}^t = \mu \Delta w_{ij}^{t-1} + 0.1 \left( \langle p_i p_j \rangle_{Q_t^0} - \langle p_i p_j \rangle_{Q_t^1} - 0.0001 w_{ij}^t \right) \qquad (6)$$

where $Q_t^0$ and $Q_t^1$ are averages over the data or the one-step reconstructions for minibatch $t$, and the momentum, $\mu$, was 0 for the first 50 weight changes and 0.9 thereafter. The hidden and visible biases, $b_i$ and $b_j$, were initialized to zero. Their values were similarly altered (by treating them like connections to a unit that was always on) but with no weight decay.

Rather than picking one particular number of hidden units, we trained networks with various different numbers of units and then used discriminative performance on the validation set to decide on the most effective number of hidden units. The largest network was the best, even though each digit model contains 392,500 parameters trained on only 4,400 images. The receptive fields learned by the hidden units are quite local. Since the hidden units are fully connected and have random initial weights the learning procedure must infer the spatial proximity of pixels from the statistics of their joint activities. Figure 1 shows the mean goodness scores of all 10 models on all 10 digit classes.

Figure 2 shows reconstructions produced by the bottom-level models on previously unseen data from the digit class they were trained on and also on data from a different digit class. With 500 hidden units, the 7s model is almost perfect at reconstructing 9s. This is because a model gets better at reconstructing more or less any image as its set of available features becomes more varied and more local. Despite this, the larger networks give better discriminative information.

### 3.1 Multi-layer models

Networks that use a single layer of hidden units and do not allow connections within a layer have some major advantages over more general networks. With an image clamped on the visible units, the hidden units are conditionally independent. So it is possible to compute an unbiased sample of the binary states of the hidden units without any iteration. This property makes PoE's easy to train and it is lost in more general architectures. If, for example, we introduce a second hidden layer that is symmetrically connected to the first hidden layer, it is no longer straightforward to compute the posterior expected activity of a unit in the first hidden layer when given an image that is assumed to have been generated by the multilayer model at thermal equilibrium. The posterior distribution can be computed by alternating Gibbs sampling between the two hidden layers, but this is slow and noisy.

Fortunately, if our ultimate goal is discrimination, there is a computationally convenient alternative to using a multilayer Boltzmann machine. Having trained a one-hidden-layer PoE on a set of images, it is easy to compute the expected activities of the hidden units on each image in the training set. These hidden activity vectors will themselves have interesting statistical structure because a PoE is not attempting to find independent causes and has no implicit penalty for using hidden units that are marginally highly correlated. So we can learn a completely separate PoE model in which the activity vectors of the hidden units are treated as the observed data and a new layer of hidden units learns to model the structure of this "data". It is not entirely clear how this second-level PoE model helps as a way of modelling the original image distribution, but it is clear that if a first-level PoE is trained on images of 2's, we would expect the vectors of hidden activities to be be very different when it is presented with a 3, even if the features it has learned are quite good at reconstructing the 3. So a second-level model should be able to assign high scores to the vectors of hidden activities that are typical of the 2 model when it is given images of 2's and low scores to the hidden activities of the 2 model when it is given images that contain combinations of features that are not normally present at the same time in a 2.

We used a three-level hierachy of PoE's for each digit class. The levels were trained sequentially and to simplify the research we always used the same number of hidden units at each level. We trained models of five different sizes with 25, 100, 200, 400, and 500 hidden units per level.

## 4 The logistic classification network

An attractive aspect of PoE's is that it is easy to compute the numerator in Eq. 1 so it is easy to compute a goodness score which is equal to the log probability of a data vector up to an additive constant. Figure 3 show the goodness of the 7s and 9s models (the most difficult pair of digits to discriminate) when presented with test images of both 7s and 9s. It can be seen that a line can be passed that separates the two digit sets almost perfectly. It is also encouraging that all of the errors are close to the decision boundary, so there are no confident misclassifications.

The classification network had 10 output units, each of which computed a logit, $x$, that was a linear function of the goodness scores, $g$, of the various PoE models, $m$, on an image, $c$. The probability assigned to class $j$ was then computed by taking a "softmax" of the logits:

$$p_j^c = \frac{e^{x_j^c}}{\sum_k e^{x_k^c}} \qquad x_j^c = b_j + \sum_m g_m^c w_{mj} \qquad (7)$$

There were 10 digit classes each with a three-level hierarchy of PoE models, so the classification network had 30 inputs and therefore 300 weights and 10 output biases. Both weights and biases were initialized to zero. The weights were learned by a momentum version of

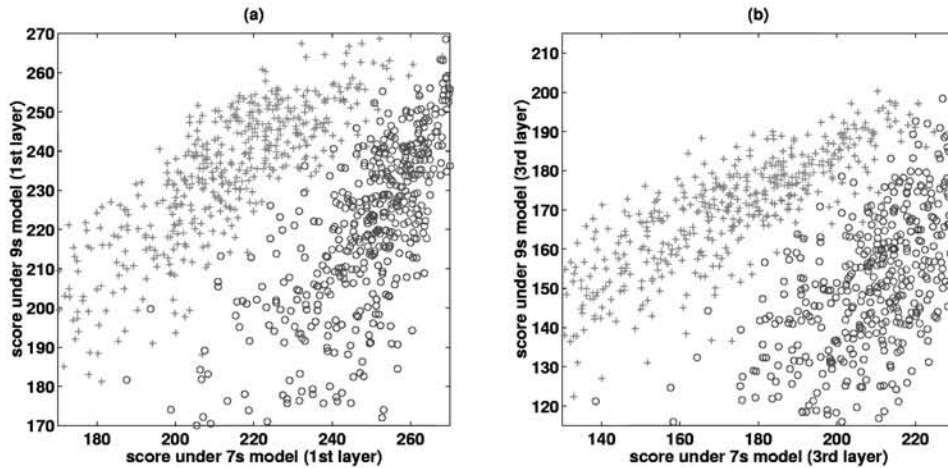

Figure 3: Validation set cross goodness results of (a) the first-level model and (b) the third-level model of 7s and 9s. All models have 500 hidden units. The third-level models clearly give higher goodness scores for second-level hidden activities in their own hierarchy than for the hidden activities in the other hierarchy.

gradient ascent in the log probability assigned to the correct class. Since there were only 310 weights to train, little effort was devoted to making the learning efficient.

$$\Delta w_{mj}(t) \;=\; \mu \Delta w_{mj}(t-1) + 0.0002 \sum_c g_m^c (t_j^c - p_j^c) \tag{8}$$

where $t_j^c$ is 1 if class $j$ is the correct answer for training case $c$ and 0 otherwise. The momentum $\mu$ was 0.9. The biases were treated as if they were weights from an input that always had a value of 1 and were learned in exactly the same way.

In each training epoch the weight changes were averaged over the whole supervised training set[1]. We used separate data for training the classification network because we expect the goodness score produced by a PoE of a given class to be worse and more variable on exemplars of that class that were not used to train the PoE and it is these poor and noisy scores that are relevant for the real, unseen test data.

The training algorithm was run using goodness scores from PoE networks with different numbers of hidden units. The results in Table 2 show a consistent improvement in classification error as the number of units in the hidden layers of each PoE increase. There is no evidence for over-fitting, even though large PoE's are very good at reconstructing images of other digit classes or the hidden activity vectors of lower-level models in other hierarchies. It is possible to reduce the error rate by a further 0.1% by averaging together the goodness scores of corresponding levels of model hierarchies with 100 or more units per layer, but this model averaging is not nearly as effective as using extra levels.

## 5   Model-based normalization

The results of our current system are still not nearly as good as human performance. In particular, it appears the network has only a very limited understanding of image invari-

| Network size | Learning epochs | % Errors |
|:---:|:---:|:---:|
| 25 | 25 | 3.8 |
| 100 | 100 | 2.3 |
| 200 | 200 | 2.2 |
| 400 | 200 | 2.0 |
| 500 | 500 | 1.7 |

Table 2: MNIST test set error rate as a function of the number of hidden units per level. There is no evidence of overfitting even when over 250,000 parameters are trained on only 4,400 examples.

ances. This is not surprising since it is trained on prenormalized data. Dealing with image invariances better will be essential for approaching human performance. The fact that we are using generative models suggests an interesting way of refining the image normalization. If the normalization of an image is slightly wrong we would expect it to have lower probability under the correct class-specific model. So we should be able to use the gradient of the goodness score to iteratively adjust the normalization so that the data fits the model better. Using $x$ translation as an example,

$$\frac{\partial G}{\partial x} = \sum_i \frac{\partial s_i}{\partial x} \frac{\partial G}{\partial s_i} \qquad \frac{\partial G}{\partial s_i} = b_i + \sum_j s_j w_{ji}$$

where $s_i$ is the intensity of pixel $i$. $\partial s_i / \partial x$ is easily computed from the intensities of the left and right neighbors of pixel $i$ and $\partial G / \partial s_i$ is just the top-down input to a pixel during reconstruction. Preliminary simulations by Yee Whye Teh on poorly normalized data show that this type of model-based renormalization improves the score of the correct model much more than the scores of the incorrect ones and thus eliminates most of the classification errors.

### Acknowledgments

We thank Yann Le Cun, Mike Revow and members of the Gatsby Unit for helpful discussions. This research was funded the Gatsby Charitable Foundation.

## References

[1] G. E. Hinton. Training products of experts by minimizing contrastive divergence. Technical Report GCNU TR 2000-004, Gatsby Computational Neuroscience Unit, University College London, 2000.

[2] P. Smolensky. Information processing in dynamical systems: Foundations of harmony theory. In D. E. Rumelhart and J. L. McClelland, editors, *Parallel Distributed Processing: Explorations in the Microstructure of Cognition. Volume 1: Foundations*. MIT Press, 1986.

[3] Yoav Freund and David Haussler. Unsupervised learning of distributions of binary vectors using 2-layer networks. In John E. Moody, Steve J. Hanson, and Richard P. Lippmann, editors, *Advances in Neural Information Processing Systems*, volume 4, pages 912–919. Morgan Kaufmann Publishers, Inc., 1992.

[4] G. E. Hinton and T. J. Sejnowski. Learning and relearning in boltzmann machines. In D. E. Rumelhart and J. L. McClelland, editors, *Parallel Distributed Processing: Explorations in the Microstructure of Cognition. Volume 1: Foundations*. MIT Press, 1986.

[5] Y. LeCun, L. D. Jackel, L. Bottou, A. Brunot, C. Cortes, J. S. Denker, H. Drucker, I. Guyon, U. A. Muller, E. Sackinger, P. Simard, and V. Vapnik. Comparison of learning algorithms for handwritten digit recognition. In F. Fogelman and P. Gallinari, editors, *International Conference on Artificial Neural Networks*, pages 53–60, Paris, 1995. EC2 & Cie.

[6] Chris J.C. Burges and B. Schölkopf. Improving the accuracy and speed of support vector machines. In Michael C. Mozer, Michael I. Jordan, and Thomas Petsche, editors, *Advances in Neural Information Processing Systems*, volume 9, page 375. The MIT Press, 1997.